# Boxlets: a Fast Convolution Algorithm for Signal Processing and Neural Networks

**Patrice Y. Simard\*, Léon Bottou, Patrick Haffner and Yann LeCun**
AT&T Labs-Research
100 Schultz Drive, Red Bank, NJ 07701-7033
patrice@microsoft.com
{leonb,haffner,yann}@research.att.com

## Abstract

Signal processing and pattern recognition algorithms make extensive use of convolution. In many cases, computational accuracy is not as important as computational speed. In feature extraction, for instance, the features of interest in a signal are usually quite distorted. This form of noise justifies some level of quantization in order to achieve faster feature extraction. Our approach consists of approximating regions of the signal with low degree polynomials, and then differentiating the resulting signals in order to obtain impulse functions (or derivatives of impulse functions). With this representation, convolution becomes extremely simple and can be implemented quite effectively. The true convolution can be recovered by integrating the result of the convolution. This method yields substantial speed up in feature extraction and is applicable to convolutional neural networks.

## 1  Introduction

In pattern recognition, convolution is an important tool because of its translation invariance properties. Feature extraction is a typical example: The distance between a small pattern (i.e. feature) is computed at all positions (i.e. translations) inside a larger one. The resulting "distance image" is typically obtained by convolving the feature template with the larger pattern. In the remainder of this paper we will use the terms image and pattern interchangeably (because of the topology implied by translation invariance).

There are many ways to convolve images efficiently. For instance, a multiplication of images of the same size in the Fourier domain corresponds to a convolution of the two images in the original space. Of course this requires $KN \log N$ operations (where $N$ is the number of pixels of the image and $K$ is a constant) just to go in and out of the Fourier domain. These methods are usually not appropriate for feature extraction because the feature to be extracted is small with respect to the image. For instance, if the image and the feature have respectively $32 \times 32$ and $5 \times 5$ pixels,

the full convolution can be done in $25 \times 1024$ multiply-adds. In contrast, it would require $2 \times K \times 1024 \times 10$ to go in and out of the Fourier domain.

Fortunately, in most pattern recognition applications, the interesting features are already quite distorted when they appear in real images. Because of this inherent noise, the feature extraction process can usually be approximated (to a certain degree) without affecting the performance. For example, the result of the convolution is often quantized or thresholded to yield the presence and location of distinctive features [1]. Because precision is typically not critical at this stage (features are rarely optimal, thresholding is a crude operation), it is often possible to quantize the signals before the convolution with negligible degradation of performance.

The subtlety lies in choosing a quantization scheme which can speed up the convolution while maintaining the same level of performance. We now introduce the convolution algorithm, from which we will deduce the constraints it imposes on quantization.

The main algorithm introduced in this paper is based on a fundamental property of convolutions. Assuming that $f$ and $g$ have finite support and that $f^n$ denotes the $n$-th integral of $f$ (or the $n$-th derivative if $n$ is negative), we can write the following convolution identity:

$$(f * g)^n = f^n * g = f * g^n \tag{1}$$

where $*$ denotes the convolution operator. Note that $f$ or $g$ are not necessarily differentiable. For instance, the impulse function (also called Dirac delta function), denoted $\delta$, verifies the identity:

$$\delta_a^n * \delta_b^m = \delta_{a+b}^{m+n} \tag{2}$$

where $\delta_a^n$ denotes the $n$-th integral of the delta function, translated by $a$ ($\delta_a(x) = \delta(x - a)$). Equations 1 and 2 are not new to signal processing. Heckbert has developed an effective filtering algorithm [2] where the filter $g$ is a simple combination of polynomial of degree $n - 1$. Convolution between a signal $f$ and the filter $g$ can be written as

$$f * g = f^n * g^{-n} \tag{3}$$

where $f^n$ is the $n$-th integral of the signal, and the $n$-th derivative of the filter $g$ can be written exclusively with delta functions (resulting from differentiating $n - 1$ degree polynomials $n$ times). Since convolving with an impulse function is a trivial operation, the computation of Equation 3 can be carried out effectively. Unfortunately, Heckbert's algorithm is limited to simple polynomial filters and is only interesting when the filter is wide and when the Fourier transform is unavailable (such as in variable length filters).

In contrast, in feature extraction, we are interested in small and arbitrary filters (the features). Under these conditions, the key to fast convolution is to quantize the images to combinations of low degree polynomials, which are differentiated, convolved and then integrated. The algorithm is summarized by equation:

$$f * g \approx F * G = (F^{-n} * G^{-m})^{m+n} \tag{4}$$

where $F$ and $G$ are polynomial approximation of $f$ and $g$, such that $F^{-n}$ and $G^{-m}$ can be written as sums of impulse functions and their derivatives. Since the convolution $F^{-n} * G^{-m}$ only involves applying Equation 2, it can be computed quite effectively. The computation of the convolution is illustrated in Figure 1. Let $f$ and $g$ be two arbitrary 1-dimensional signals (top of the figure). Let's assume that $f$ and $g$ can both be approximated by partitions of polynomials, $F$ and $G$. On the figure, the polynomials are of degree 0 (they are constant), and are depicted in the second line. The details on how to compute $F$ and $G$ will be explained in the next section. In the next step, $F$ and $G$ are differentiated once, yielding successions of impulse functions (third line in the figure). The impulse representation has the advantage of having a finite support, and of being easy to convolve. Indeed two impulse functions can be convolved using Equation 2 ($4 \times 3 = 12$ multiply-adds on the figure). Finally the result of the convolution must be integrated twice to yield

$$F * G = (F^{-1} * G^{-1})^2 \tag{5}$$

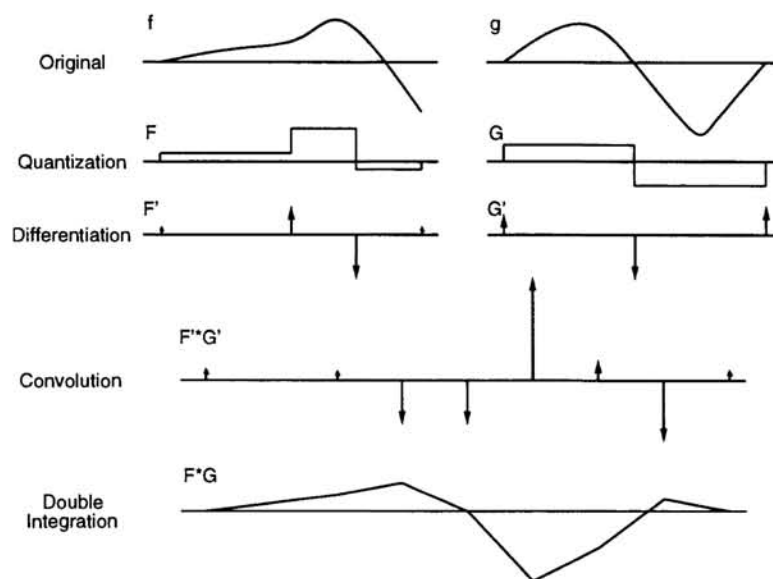

Figure 1: Example of convolution between 1-dimensional function $f$ and $g$, where the approximations of $f$ and $g$ are piecewise constant.

## 2 Quantization: from Images to Boxlets

The goal of this section is to suggest efficient ways to approximate an image $f$ by cover of polynomials of degree $d$ suited for convolution. Let $S$ be the space on which $f$ is defined, and let $C = \{c_i\}$ be a partition of $S$ ($c_i \bigcap c_j = \emptyset$ for $i \neq j$, and $\bigcup_i c_i = S$). For each $c_i$, let $p_i$ be a polynomial of degree $d$ which minimizes equation:

$$e_i = \int_{x \in c_i} (f(x) - p_i(x))^2 dx \tag{6}$$

The uniqueness of $p_i$ is guaranteed if $c_i$ is convex. The problem is to find a cover $C$ which minimizes both the number of $c_i$ and $\sum_i e_i$. Many different compromises are possible, but since the computational cost of the convolution is proportional to the number of regions, it seemed reasonable to chose the largest regions with a *maximum error* bounded by a threshold $K$. Since each region will be differentiated and integrated along the directions of the axes, the boundaries of the $c_i$s are restricted to be parallel to the axes, hence the appellation *boxlet*. There are still many ways to compute valid partitions of boxlets and polynomials. We have investigated two very different approaches which both yield a polynomial cover of the image in reasonable time. The first algorithm is greedy. It uses a procedure which, starting from a top left corner, finds the biggest boxlet $c_i$ which satisfies $e_i < K$ without overlapping another boxlet. The algorithm starts with the top left corner of the image, and keeps a list of all possible starting points (uncovered top left corners) sorted by X and Y positions. When the list is exhausted, the algorithm terminates. Surprisingly, this algorithm can run in $O(d(N + P \log N))$, where $N$ is the number of pixels, $P$ is the number of boxlets and $d$ is the order of the polynomials $p_i$s. Another much simpler algorithm consists of recursively splitting boxlets, starting from a boxlet which encompass the whole image, until $e_i < K$ for all the leaves of the tree. This algorithm runs in $O(dN)$, is much easier to implement, and is faster (better time constant). Furthermore, even though the first algorithm yields a polynomial coverage with less boxlets, the second algorithm yields less impulse functions after differentiation because more impulse functions can be combined (see next section). Both algorithms rely on the fact that Equation 6 can be computed

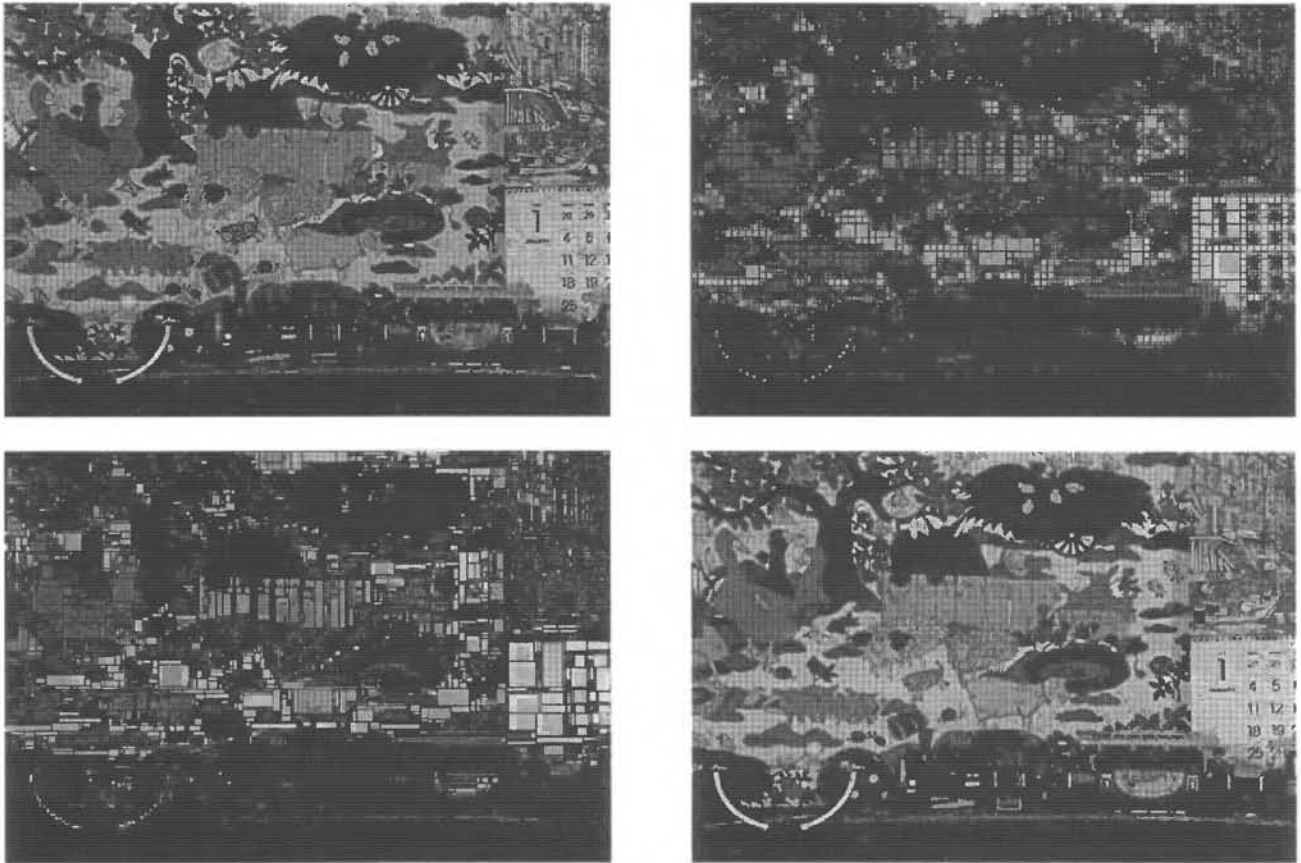

Figure 2: *Effects of boxletization: original (top left), greedy (bottom left) with a threshold of 10,000, and recursive (top and bottom right) with a threshold of 10,000.*

in constant time. This computation requires the following quantities

$$\underbrace{\sum f(x,y), \sum f(x,y)^2}_{\text{degree 0}}, \underbrace{\sum f(x,y)x, \sum f(x,y)y, \sum f(x,y)xy}_{\text{degree 1}}, \ldots \quad (7)$$

to be pre-computed over the whole image, for the greedy algorithm, or over recursively embedded regions, for the recursive algorithm. In the case of the recursive algorithm these quantities are computed bottom up and very efficiently. To prevent the sums to become too large a limit can be imposed on the maximum size of $c_i$. The coefficients of the polynomials are quickly evaluated by solving a small linear system using the first two sums for polynomials of degree 0 (constants), the first 5 sums for polynomials of degree 1, and so on.

Figure 2 illustrates the results of the quantization algorithms. The top left corner is a fraction of the original image. The bottom left image illustrates the boxletization of the greedy algorithm, with polynomials of degree 1, and $e_i <= 10,000$ ( 13000 boxlets, 62000 impulse (and its derivative) functions. The top right image illustrates the boxletization of the recursive algorithm, with polynomials of degree 0 and $e_i <= 10,000$ ( 47000 boxlets, 58000 impulse functions). The bottom right is the same as top right without displaying the boxlet boundaries. In this case the pixel to impulse function ratio 5.8.

## 3  Differentiation: from Boxlets to Impulse Functions

If $p_i$ is a polynomial of degree $d$, its $(d+1)$-th derivative can be written as a sum of impulse function's derivatives, which are zero everywhere but at the corners of $c_i$. These impulse functions summarize the boundary conditions and completely characterize $p_i$. They can be represented by four $(d+1)$-dimensional vectors associated with the 4 corners of $c_i$. Figure 3 (top) illustrates the impulse functions at the 4

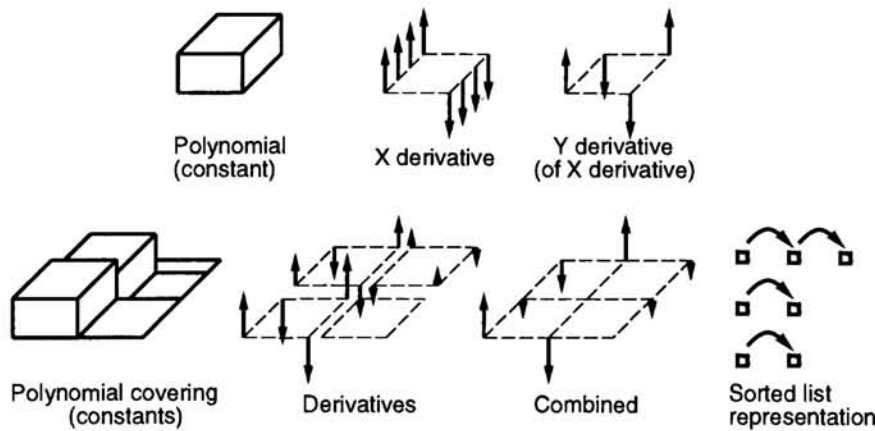

Figure 3: *Differentiation of a constant polynomial in 2D (top). Combining the derivative of adjacent polynomials (bottom)*

corners when the polynomial is a constant (degree zero). Note that the polynomial must be differentiated $d + 1$ times (in this example the polynomial is a constant, so $d = 0$), with respect to each dimension of the input space. This is illustrated at the top of Figure 3. The cover $C$ being a partition, boundary conditions between adjacent squares do simplify, that is, the same derivatives of a impulse functions at the same location can be combined by adding their coefficients. It is very advantageous to do so because it will reduce the computation of the convolution in the next step. This is illustrated in Figure 3 (bottom). This combining of impulse functions is one of the reason why the recursive algorithm for the quantization is preferred to the greedy algorithm. In the recursive algorithm, the boundaries of boxlets are often aligned, so that the impulse functions of adjacent boxlets can be combined. Typically, after simplification, there are only 20% more impulse functions than there are boxlets. In contrast, the greedy algorithm generates up to 60% more impulse functions than boxlets, due to the fact that there are no alignment constraints. For the same threshold the recursive algorithm generates 20% to 30% less impulse functions than the greedy algorithm.

Finding which impulse functions can be combined is a difficult task because the recursive representation returned by the recursive algorithm does not provide any means for matching the bottom of squares on one line, with the top of squares from below that line. Sorting takes $O(P \log P)$ computational steps (where $P$ is the number of impulse functions) and is therefore too expensive. A better algorithm is to visit the recursive tree and accumulate all the top corners into sorted (horizontal) lists. A similar procedure sorts all the bottom corners (also into horizontal lists). The horizontal lists corresponding to the same vertical positions can then be merged in $O(P)$ operations. The complete algorithm which quantizes an image of $N$ pixels and returns sorted lists of impulse functions runs in $O(dN)$ (where $d$ is the degree of the polynomials).

## 4 Results

The convolution speed of the algorithm was tested with feature extraction on the image shown on the top left of Figure 2. The image is quantized, but the feature is not. The feature is tabulated in kernels of sizes $5 \times 5$, $10 \times 10$, $15 \times 15$ and $20 \times 20$. If the kernel is decomposable, the algorithm can be modified to do two 1D convolutions instead of the present 2D convolution.

The quantization of the image is done with constant polynomials, and with thresholds varying from 1,000 to 40,000. This corresponds to varying the pixel to impulse function ratio from 2.3 to 13.7. Since the feature is not quantized, these ratios correspond exactly to the ratios of number of multiply-adds for the standard convolution versus the boxlet convolution (excluding quantization and integration). The

| Threshold | Image | | | Convolution kernel size | | | |
|---|---|---|---|---|---|---|---|
| | Boxlets | Impls f. | Ratio | 5x5 | 10x10 | 15x15 | 20x20 |
| 1,000 | 125,685 | 144,520 | 2.3 | 1.5 | 2.2 | 2.4 | 2.4 |
| | | | | 2.3 | 2.6 | 2.6 | 2.5 |
| 5,000 | 68,994 | 84.382 | 4.0 | 2.3 | 3.2 | 3.8 | 4.0 |
| | | | | 3.8 | 3.8 | 4.0 | 4.0 |
| 10,000 | 47,253 | 58,120 | 5.8 | 2.8 | 4.8 | 5.4 | 5.5 |
| | | | | 4.7 | 6.0 | 6.1 | 5.9 |
| 40,000 | 20,244 | 24,661 | 13.7 | 5.2 | 9.2 | 11.3 | 12.4 |
| | | | | 8.4 | 12.5 | 13.4 | 13.8 |

Table 1: *Convolution speed-up factors*

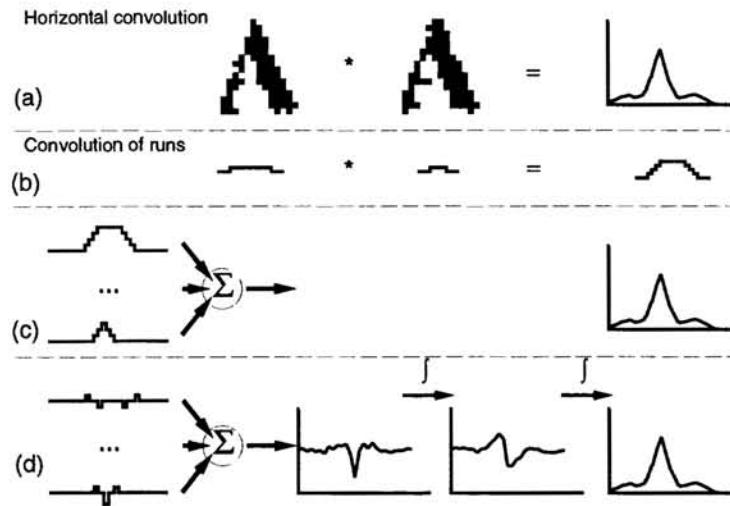

Figure 4: *Run length X convolution*

actual speed up factors are summarized in Table 1. The four last columns indicate the measured time ratios between the standard convolution and the boxlet convolution. For each threshold value, the top line indicates the time ratio of standard convolution versus quantization, convolution and integration time for the boxlet convolution. The bottom line does not take into account the quantization time. The feature size was varied from $5 \times 5$ to $20 \times 20$. Thus with a threshold of 10,000 and a $5 \times 5$ kernel, the quantization ratio is 5.8, and the speed up factor is 2.8. The loss in image quality can be seen by comparing the top left and the bottom right images. If several features are extracted, the quantization time of the image is shared amongst the features and the speed up factor is closer to 4.7.

It should be noted that these speed up factors depend on the quantization level which depends on the data and affects the accuracy of the result. The good news is that for each application the optimal threshold (the maximum level of quantization which has negligible effect on the result) can be evaluated quickly. Once the optimal threshold has been determined, one can enjoy the speed up factor. It is remarkable that with a quantization factor as low as 2.3, the speed up ratio can range from 1.5 to 2.3, depending on the number of features. We believe that this method is directly applicable to forward propagation in convolutional neural nets (although no results are available at this time).

The next application shows a case where quantization has no adverse effect on the accuracy of the convolution, and yet large speed ups are obtained.

## 5   Binary images and run-length encoding

The quantization steps described in Sections 2 and 3 become particularly simple when the image is binary. If the threshold is set to zero, and if only the X derivative is considered, the impulse representation is equivalent to run-length encoding. Indeed the position of each positive impulse function codes the beginning of a run, while the position of each negative impulses code the end of a run. The horizontal convolution can be computed effectively using the boxlet convolution algorithm. This is illustrated in Figure 4. In (a), the distance between two binary images must be evaluated for every horizontal position (horizontal translation invariant distance). The result is obtained by convolving each horizontal line and by computing the sum of each of the convolution functions. The convolution of two runs, is depicted in (b), while the summation of all the convolutions of two runs is depicted in (c). If an impulse representation is used for the runs (a first derivative), each summation of a convolution between two runs requires only 4 additions of impulse functions, as depicted in (d). The result must be integrated twice, according to Equation 5. The speed up factors can be considerable depending on the width of the images (an order of magnitude if the width is 40 pixels), and there is no accuracy penalty.

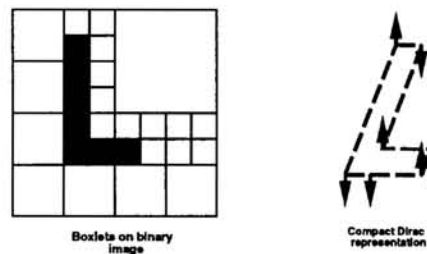

Boxlets on binary image          Compact Dirac representation

Figure 5: Binary image (left) and compact impulse function encoding (right).

This speed up also generalizes to 2-dimensional encoding of binary images. The gain comes from the frequent cancellations of impulse functions of adjacent boxlets. The number of impulse functions is proportional to the contour length of the binary shapes. In this case, the boxlet computation is mostly an efficient algorithm for 2-dimensional run-length encoding. This is illustrated in Figure 5. As with run-length encoding, a considerable speed up is obtained for convolution, at no accuracy penalty cost.

## 6   Conclusion

When convolutions are used for feature extraction, precision can often be sacrificed for speed with negligible degradation of performance. The boxlet convolution method combines quantization and convolution to offer a continuous adjustable trade-off between accuracy and speed. In some cases (such as in relatively simple binary images) large speed ups can come with no adverse effects. The algorithm is directly applicable to the forward propagation in convolutional neural networks and in pattern matching when translation invariance results from the use of convolution.

## Footnotes

\* Now with Microsoft, One Microsoft Way, Redmond, WA 98052

## References

[1] Yann LeCun and Yoshua Bengio, "Convolutional networks for images, speech, and time-series," in *The Handbook of Brain Theory and Neural Networks*, M. A. Arbib, Ed. 1995, MIT Press.

[2] Paul S. Heckbert, "Filtering by repeated integration," in *ACM SIGGRAPH conference on Computer graphics*, Dallas, TX, August 1986, vol. 20, pp. 315–321.
